# Response Analysis of Neuronal Population with Synaptic Depression

**Wentao Huang**
Institute of Intelligent Information
Processing, Xidian University,
Xi'an 710071, China
wthuang@mail.xidian.edu.cn

**Licheng Jiao**
Institute of Intelligent Information
Processing, Xidian University,
Xi'an 710071, China
lchjiao@mail.xidian.edu.cn

**Shan Tan**
Institute of Intelligent Information
Processing, Xidian University,
Xi'an 710071, China
shtan@mail.xidian.edu.cn

**Maoguo Gong**
Institute of Intelligent Information
Processing, Xidian University,
Xi'an 710071, China
mggong@mail.xidian.edu.cn

## Abstract

In this paper, we aim at analyzing the characteristic of neuronal population responses to instantaneous or time-dependent inputs and the role of synapses in neural information processing. We have derived an evolution equation of the membrane potential density function with synaptic depression, and obtain the formulas for analytic computing the response of instantaneous fire rate. Through a technical analysis, we arrive at several significant conclusions: The background inputs play an important role in information processing and act as a switch betwee temporal integration and coincidence detection. the role of synapses can be regarded as a spatio-temporal filter; it is important in neural information processing for the spatial distribution of synapses and the spatial and temporal relation of inputs. The instantaneous input frequency can affect the response amplitude and phase delay.

## 1 Introduction

Noise has an important impact on information processing of the nervous system in vivo. It is significance for us to study the stimulus-and-response behavior of neuronal populations, especially to transients or time-dependent inputs in noisy environment, viz. given this stochastic environment, the neuronal output is typically characterized by the instantaneous firing rate. It has come in for a great deal of attention in recent years[1-4]. Moreover, it is revealed recently that synapses have a more active role in information processing[5-7]. The synapses are highly dynamic and show use-dependent plasticity over a wide range of time scales. Synaptic short-term depression is one of the most common expressions of plasticity. At synapses with this type of modulation, pre-synaptic activity produces a decrease in synaptic. The present work is concerned with the processes underlying investigating the collectivity dynamics of neuronal population with synaptic depression and

the instantaneous response to time-dependence inputs. First, we deduce a one-dimension Fokker-Planck (FP) equation via reducing the high-dimension FP equations. Then, we derive the stationary solution and the response of instantaneous fire rate from it. Finally, the models are analyzed and discussed in theory and some conclusions are presented.

## 2 Models and Methods

### 2.1 Single Neuron Models and Density Evolution Equations

Our approach is based on the integrate-and-fire(IF) neurons. The population density based on the integrate-and-fire neuronal model is low-dimensional and thus can be computed efficiently, although the approach could be generalized to other neuron models. It is completely characterized by its membrane potential below threshold. Details of the generation of an action potential above the threshold are ignored. Synaptic and external inputs are summed until it reaches a threshold where a spike is emitted. The general form of the dynamics of the membrane potential $v$ in IF model can be written as

$$\tau_v \frac{dv(t)}{dt} = -v(t) + S_e(t) + \tau_v \sum_{k=1}^{N} J_k(t)\delta(t - t_k^{sp}), \tag{1}$$

where $0 \le v \le 1$, $\tau_v$ is the membrane time constant, $S_e(t)$ is an external current directly injected in the neuron, $N$ is the number of synaptic connections, $t_k^{sp}$ is occurring time of the firing of a presynaptic neuron $k$ and obeys a Poisson distribution with mean $\lambda_k$, $J_k(t)$ is the efficacy of synapse $k$. The transmembrane potential, $v$, has been normalized so that $v = 0$ marks the rest state, and $v = 1$ the threshold for firing. When the latter is achieved, $v$ is reset to zero. $J_k(t) = AD_k(t)$, where $A$ is a constant representing the absolute synaptic efficacy corresponding to the maximal postsynaptic response obtained if all the synaptic resources are released at once, and $D_k(t)$ act in accordance with complex dynamics rule. We use the phenomenological model by Tsodyks & Markram [7] to simulate short-term synaptic depression:

$$\frac{dD_k(t)}{dt} = \frac{(1 - D_k(t))}{\tau_d} - U_k D_k(t)\delta(t - t_k^{sp}), \tag{2}$$

where $D_k$ is a 'depression' variable, $D_k \in [0, 1]$, $\tau_d$ is the recovery time constant, $U_k$ is a constant determining the step decrease in $D_k$. Using the diffusion approximation, we can get from (1) and (2)

$$\tau_v \frac{dv(t)}{dt} = -v(t) + S_e(t) + \tau_v \sum_{k=1}^{N} AD_k(\lambda_k + \sqrt{\lambda_k}\xi_k(t)),$$

$$\frac{dD_k(t)}{dt} = \frac{(1 - D_k)}{\tau_d} - U_k D_k(\lambda_k + \sqrt{\lambda_k}\xi_k(t)). \tag{3}$$

The Fokker-Planck equation of equations (3) is

$$\frac{\partial p(t, v, \mathbf{D})}{\partial t} = -\frac{\partial}{\partial v}\left(\frac{-v + K_v}{\tau_v}p\right) - \sum_{k=1}^{N}\frac{\partial}{\partial D_k}(K_{D_k}p) - \sum_{k=1}^{N}\frac{\partial}{\partial v\partial D_k}(\lambda_k AU_k D_k^2 p)$$

$$+ \frac{1}{2}\left\{\frac{\partial^2}{\partial v^2}\left(\sum_{k=1}^{N}\lambda_k A^2 D_k^2 p\right) + \sum_{k=1}^{N}\frac{\partial^2}{\partial D_k^2}(\lambda_k U_k^2 D_k^2 p)\right\},$$

$$K_v = S_e + \sum_{k=1}^{N}\tau_v\lambda_k AD_k, \qquad K_{D_k} = \frac{(1 - D_k)}{\tau_d} - \lambda_k U_k D_k. \tag{4}$$

where $\mathbf{D} = (D_1, D_2, ...D_N)$, and

$$p(t, v, \mathbf{D}) = p_d(t, \mathbf{D}|v)p_v(t, v), \qquad \int_{-\infty}^{\infty} p_d(t, \mathbf{D}|v)d\mathbf{D} = 1. \tag{5}$$

We assume that $D_1, D_2, ...D_N$ are uncorrelated, then we have

$$p_d(t, \mathbf{D}|v) = \prod_{k=1}^{N} \tilde{p}_d^k(t, D_k|v), \tag{6}$$

where $\tilde{p}_d^k(t, D_k|v)$ is the conditional probability density. Moreover, we can assume

$$\tilde{p}_d^k(t, D_k|v) \approx p_d^k(t, D_k). \tag{7}$$

Substituting (5) into (4), we get

$$p_d \frac{\partial p_v}{\partial t} + p_v \frac{\partial p_d}{\partial t} = -\frac{\partial}{\partial v}(\frac{-v + K_v}{\tau_v} p_v p_d) -$$

$$\sum_{k=1}^{N} p_v \frac{\partial}{\partial D_k}(K_{D_k} p_d) - \sum_{k=1}^{N} \frac{\partial}{\partial v \partial D_k}(AU_k D_k^2 \lambda_k p_v p_d) +$$

$$\frac{1}{2}\{\frac{\partial^2}{\partial v^2}(\sum_{k=1}^{N} \lambda_k A^2 D_k^2 p_v p_d) + \sum_{k=1}^{N} \frac{\partial^2}{\partial D_k^2}(\lambda_k U_k^2 D_k^2 p_v p_d)\}. \tag{8}$$

Integrating Eqation (8) over $\mathbf{D}$, we get

$$\tau_v \frac{\partial p_v(t, v)}{\partial t} = -\frac{\partial}{\partial v}(-v + \tilde{K}_v)p_v(t, v) + \frac{Q_v}{2} \frac{\partial^2 p_v(t, v)}{\partial v^2}, \tag{9}$$

where

$$\tilde{K}_v = \int K_v p_d d\mathbf{D} = S_e + \sum_{k=1}^{N} \tau_v \lambda_k A m_k, \; Q_v = \sum_{k=1}^{N} \tau_v \lambda_k A^2 \gamma_k,$$

$$m_k = \int D_k p_d^k(t, D_k)dD_k, \qquad \gamma_k = \int D_k^2 p_d^k(t, D_k)dD_k, \tag{10}$$

and $p_d^k(t, D_k)$ satisfies the following equation Fokker-Planck equation

$$\frac{\partial p_d^k}{\partial t} = -\frac{\partial}{\partial D_k}(K_{D_k} p_d^k) + \frac{1}{2} \frac{\partial^2}{\partial D_k^2}(U_k^2 D_k^2 \lambda_k p_d^k). \tag{11}$$

From (10) and (11), we can get

$$\frac{dm_k}{dt} = -(\frac{1}{\tau_d} + U\lambda_k)m_k + \frac{1}{\tau_d},$$

$$\frac{d\gamma_k}{dt} = -(\frac{2}{\tau_d} + (2U - U^2)\lambda_k)\gamma_k + \frac{2m_k}{\tau_d}. \tag{12}$$

Let

$$J_v(t, v) = (\frac{-v + \tilde{K}_v}{\tau_v})p_v(t, v) - \frac{Q_v}{2\tau_v} \frac{\partial p_v(t, v)}{\partial v},$$

$$r(t) = J_v(t, 1), \tag{13}$$

where $J_v(t, v)$ is the probability flux of $p_v$, $r(t)$ is the fire rate. The boundary conditions of equation (9) are

$$p_v(t, 1) = 0, \quad \int_0^1 p_v(t, v)dv = 1, \quad r(t) = J_v(t, 0). \tag{14}$$

## 2.2 Stationary Solution and Response Analysis

When the system is in the stationary states, $\partial p_v / \partial t = 0$, $dm_k / dt = 0$, $d\gamma_k / dt = 0$, $p_v(t, v) = p_v^0(v)$, $r(t) = r_0$, $m_k(t) = m_k^0$, $\gamma_k(t) = \gamma_k^0$ and $\lambda_k(t) = \lambda_k^0$. are time-independent. From (9), (12), (13) and (14), we get

$$p_v^0(v) = \frac{2\tau_v r_0}{Q_v^0} \exp[-\frac{(v - \tilde{K}_v^0)^2}{Q_v^0}] \int_v^1 \exp[\frac{(v' - \tilde{K}_v^0)^2}{Q_v^0}] dv', 0 \leq v \leq 1,$$

$$r_0 = \left( \tau_v \sqrt{\pi} \int_{\frac{-\tilde{K}_v^0}{\sqrt{Q_v^0}}}^{\frac{1-\tilde{K}_v^0}{\sqrt{Q_v^0}}} \exp(u^2)[\mathrm{erf}(\frac{\tilde{K}_v^0}{\sqrt{Q_v^0}}) + \mathrm{erf}(u)] du \right)^{-1},$$

$$\tilde{K}_v^0 = S_e + \sum_{k=1}^{N} \tau_v A \lambda_k^0 m_k^0, \qquad Q_v^0 = \sum_{k=1}^{N} \tau_v A^2 \lambda_k^0 \gamma_k^0,$$

$$m_k^0 = \frac{1}{1 + U_k \tau_d \lambda_k^0}, \qquad \gamma_k^0 = \frac{2m_k^0}{2 + \tau_d(2U_k - U_k^2)\lambda_k^0}. \tag{15}$$

Sometimes, we are more interested in the instantaneous response to time-dependence random fluctuation inputs. The inputs take the form:

$$\lambda_k = \lambda_k^0 (1 + \varepsilon_k \lambda_k^1(t)), \tag{16}$$

where $\varepsilon_k \ll 1$. Then $m_k$ and $\gamma_k$ have the forms, i.e.,

$$m_k = m_k^0 (1 + \varepsilon_k m_k^1(t) + O(\varepsilon_k^2)),$$
$$\gamma_k = \gamma_k^0 (1 + \varepsilon_k \gamma_k^1(t) + O(\varepsilon_k^2)), \tag{17}$$

and $\tilde{K}_v$ and $Q_v$ are

$$\tilde{K}_v = S_e + \sum_{k=1}^{N} \tau_v A \lambda_k^0 m_k^0 + \sum_{k=1}^{N} \varepsilon_k \tau_v A \lambda_k^0 m_k^0 (\lambda_k^1 + m_k^1)) + O(\varepsilon_k^2),$$

$$Q_v = \sum_{k=1}^{N} \tau_v A^2 \lambda_k^0 \gamma_k^0 + \sum_{k=1}^{N} \varepsilon_k \tau_v A^2 \lambda_k^0 \gamma_k^0 (\lambda_k^1 + \gamma_k^1) + O(\varepsilon_k^2). \tag{18}$$

Substituting (17) into (12), and ignoring the high order item, it yields:

$$\frac{dm_k^1}{dt} = -(\frac{1}{\tau_d} + U_k \lambda_k^0) m_k^1 - U_k \lambda_k^0 \lambda_k^1(t),$$

$$\frac{d\gamma_k^1}{dt} = -(\frac{2}{\tau_d} + (2U_k - U_k^2)\lambda_k^0)\gamma_k^1 + \frac{2m_k^1}{\tau_d} - (2U_k - U_k^2)\lambda_k^0 \lambda_k^1(t). \tag{19}$$

With the definitions

$$\tilde{K}_v = \tilde{K}_v^0 + \epsilon \tilde{K}_v^1(t) + O(\epsilon^2),$$
$$Q_v = Q_v^0 + \epsilon Q_v^1(t) + O(\epsilon^2),$$
$$p_v = p_v^0 + \epsilon p_1(t) + O(\epsilon^2),$$
$$r = r_0 + \epsilon r_1(t) + O(\epsilon^2), \tag{20}$$

where $\epsilon \ll 1$, and boundary conditions of $p_1$

$$p_1(t, 1) = 0, \qquad \int_0^1 p_1(t, v) dv = 0, \tag{21}$$

using the perturbative expansion in powers of $\epsilon$, we can get

$$0 = -\frac{\partial}{\partial v}(-v + \tilde{K}_v^0)p_v^0(v) + \frac{Q_v}{2}\frac{\partial^2 p_v^0(v)}{\partial v^2},$$

$$\tau_v \frac{\partial p_1}{\partial t} = -\frac{\partial}{\partial v}(-v + \tilde{K}_v^0)p_1 + \frac{Q_v^0}{2}\frac{\partial^2 p_1}{\partial v^2} - \frac{\partial f_0(t,v)}{\partial v},$$

$$f_0(t,v) = \tilde{K}_v^1(t)p_v^0 - \frac{Q_v^1(t)}{2}\frac{\partial p_v^0}{\partial v},$$

$$r_1 = -\frac{Q_v^0}{2\tau_v}\frac{\partial p_1(t,1)}{\partial v} - \frac{Q_v^1(t)}{2\tau_v}\frac{\partial p_v^0(1)}{\partial v}. \tag{22}$$

For the oscillatory inputs $\tilde{K}_v^1(t) = k(\omega)e^{j\omega t}$, $Q_v^1(t) = q(\omega)e^{j\omega t}$, the output has the same frequency and takes the forms $p_1(t,v) = p_\omega(\omega,v)e^{j\omega t}$, $\partial p_1/\partial t = j\omega p_1$.

For inputs that vary on a slow enough time scale, satisfy $\tau_v \omega \ll 1$, we define

$$\epsilon_l = \tau_v \omega,$$

$$p_1 = p_1^0 + \epsilon_l p_1^1 + O(\epsilon_l^2),$$

$$r_1 = r_1^0 + \epsilon_l r_1^1 + O(\epsilon_l^2). \tag{23}$$

Using the perturbative expansion in powers of $\epsilon_l$, we get

$$\frac{\partial f_0(t,v)}{\partial v} = -\frac{\partial}{\partial v}(-v + \tilde{K}_v^0)p_1^0 + \frac{Q_v^0}{2}\frac{\partial^2 p_1^0}{\partial v^2},$$

$$jp_1^0 = -\frac{\partial}{\partial v}(-v + \tilde{K}_v^0)p_1^1 + \frac{Q_v^0}{2}\frac{\partial^2 p_1^1}{\partial v^2}. \tag{24}$$

The solutions of equtions (24) are

$$p_1^n = \frac{2}{Q_v^0}\exp[-\frac{(v - \tilde{K}_v^0)^2}{Q_v^0}]\int_v^1 (\tau_v r_1^n - F_n)\exp[\frac{(v' - \tilde{K}_v^0)^2}{Q_v^0}]dv',$$

$$r_1^n = \frac{2r_0}{Q_v^0}\int_0^1 \exp[-\frac{(v - \tilde{K}_v^0)^2}{Q_v^0}]\int_v^1 F_n \exp[\frac{(v' - \tilde{K}_v^0)^2}{Q_v^0}]dv' dv,$$

$$F_0 = f_0(t,v), \qquad F_1 = j\int_0^v p_1^0(v')dv', \qquad n = 0,1. \tag{25}$$

In general, $Q_v^1(t) \ll \tilde{K}_v^1(t)$, then we have

$$F_0 = f_0(t,v) \approx \tilde{K}_v^1(t)p_v^0. \tag{26}$$

From (23), (25) and (26), we can get

$$r_1 \approx \frac{2r_0}{Q_v^0}\tilde{K}_v^1(t)\int_0^1 \exp[-\frac{(v - \tilde{K}_v^0)^2}{Q_v^0}]\int_v^1 p_v^0 \exp[\frac{(v' - \tilde{K}_v^0)^2}{Q_v^0}]dv' dv + j\omega\tau_v \times$$

$$\frac{2r_0}{Q_v^0}\int_0^1 \exp[-\frac{(v - \tilde{K}_v^0)^2}{Q_v^0}]\int_v^1 [\int_0^{v'} p_1^0(v'')dv'']\exp[\frac{(v' - \tilde{K}_v^0)^2}{Q_v^0}]dv' dv. \tag{27}$$

In the limit of high frequency inputs, i.e. $1/\tau_v\omega \ll 1$, with the definitions

$$\epsilon_h = \frac{1}{\tau_v\omega},$$

$$p_1 = p_h^0 + \epsilon_h p_h^1 + O(\epsilon_h^2), \tag{28}$$

we obtain

$$p_h^0 = 0, \qquad\qquad p_h^1 = j\frac{\partial f_0(t,v)}{\partial v},$$

$$r_1 = -\frac{Q_v^1(t)}{2\tau_v}\frac{\partial p_v^0(1)}{\partial v} - j\epsilon_h\frac{Q_v^0}{2\tau_v}\frac{\partial^2 f_0(t,1)}{\partial v^2} + O(\epsilon_h^2)$$

$$\approx \frac{Q_v^1(t)}{Q0}r_0 - j\epsilon_h\frac{Q_v^0}{2\tau_v}(\tilde{K}_v^1(t)\frac{\partial^2 p_v^0(1)}{\partial v^2} - \frac{Q_v^1(t)}{2}\frac{\partial^3 p_v^0}{\partial v^3})$$

$$= \frac{Q_v^1(t)r_0}{Q_v^0} - \frac{2j\epsilon_h\tilde{K}_v^1(t)r_0}{Q_v^0}\left((1-\tilde{K}_v^0) - \frac{Q_v^1(t)}{\tilde{K}_v^1(t)Q_v^0}\left(1-\tilde{K}_v^0 - Q_v^0\right)\right). \qquad (29)$$

When $Q_v^1(t) \ll \tilde{K}_v^1(t)$, we have

$$r_1 \approx \frac{Q_v^1(t)r_0}{Q_v^0} - \frac{2j\tilde{K}_v^1(t)r_0}{\tau_v\omega Q_v^0}(1-\tilde{K}_v^0)(1-\frac{Q_v^1(t)}{\tilde{K}_v^1(t)Q_v^0}), \qquad (30)$$

## 3  Discussion

In equation (15), $\tilde{K}_v^0$ reflects the average intensity of background inputs and $Q_v^0$ reflects the intensity of background noise. When $1 \ll \tau_d U_k\lambda_k^0$, we have

$$\tilde{K}_v^0 \approx S_e + \sum_{k=1}^{N}\frac{\tau_v A}{\tau_d U_k},$$

$$Q_v^0 \approx \sum_{k=1}^{N}\frac{\tau_v A^2}{\tau_d U_k(1 + \tau_d U_k\lambda_k^0(1-U_k/2))}. \qquad (31)$$

From (31), we can know the change of background inputs $\lambda_k^0$ has little influence on $\tilde{K}_v^0$ which is dominated by parameter $\tau_v A/\tau_d U_k$, but more influence on $Q_v^0$ which decreases with $\lambda_k^0$ increasing.

In the low input frequency regime, from (27), we can know that the input frequency $\omega$ increasing will result in the response amplitude and the phase delay increasing. However, in the high input frequency limit regime, from (30), we can know the input frequency $\omega$ increasing will result in the response amplitude and the phase delay decreasing. Moreover, from (27) and (30), we know the stationary background fire rate $r_0$ play an important part in response to changes in fluctuation outputs. The instantaneous response $r_1$ increases monotonically with background fire rate $r_0$.But the background fire rate $r_0$ is a function of the background noise $Q_v^0$. In equation (27), $\left\|r_1/\tilde{K}_v^1\right\|$ reflects the response amplitude, and in equation (30), $r_0/Q_v^0$ reflects the response amplitude. As Figure 1 (A) and (B) show that $\left\|r_1/\tilde{K}_v^1\right\|$ and $r_0/Q_v^0$ changes with variables $Q_v^0$ and $\tilde{K}_v^0$ respectively. We can know, for the subthreshold regime ($\tilde{K}_v^0 < 1$), they increase monotonically with $Q_v^0$ when $\tilde{K}_v^0$ is a constant. However, for the suprathreshold regime ($\tilde{K}_v^0 > 1$), they decrease monotonically with $Q_v^0$ when $\tilde{K}_v^0$ is a constant. When inputs remain, if the instantaneous response amplitude increases, then we can take for the role of neurons are more like coincidence detection than temporal integration. And from this viewpoint, it suggests that the background inputs play an important role in information processing and act as a switch between temporal integration and coincidence detection.

In equation (16), if the inputs take the oscillatory form, $\lambda_k^1(t) = e^{j\omega t}$, according to (19),

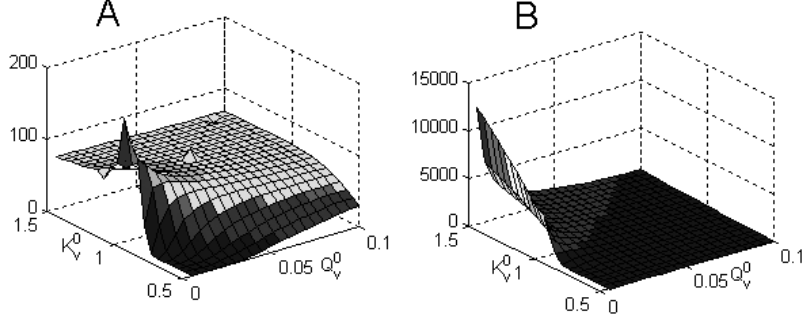

Figure 1: Response amplitude versus $Q_v^0$ and $\widetilde{K}_v^0$. (A) $\left\|r_1/\tilde{K}_v^1\right\|$ (for equation (27)) changes with $Q_v^0$ and $\tilde{K}_v^0$. (B) $r_0/Q_v^0$ (for equation (30)) changes with $Q_v^0$ and $\tilde{K}_v^0$.

we get

$$m_k^1 = -\frac{\tau_d U_k \lambda_k^0 e^{j(\omega t - \theta_m)}}{\sqrt{(\tau_d \omega)^2 + (1 + \tau_d U_k \lambda_k^0)^2}}, \tag{32}$$

where $\theta_m = \operatorname{arctg}(\frac{\tau_d \omega}{1 + \tau_d U_k \lambda_k^0})$ is the phase delay, $\tau_d U_k \lambda_k^0 / \sqrt{(\tau_d \omega)^2 + (1 + \tau_d U_k \lambda_k^0)^2}$ is the amplitude. The minus shows it is a 'depression' response amplitude. The phase delay increases with the input frequency $\omega$ and decreases with the background input $\lambda_k^0$. The 'depression' response amplitude decrease with the input frequency $\omega$ and increase with the background input $\lambda_k^0$. The equations (15) (18), (12), (19), (27), (30) and (32) show us a point of view that the synapses can be regarded as a time-dependent external field which impacts on the neuronal population through the time-dependent mean and variance. We assume the inputs are composed of two parts, viz. $\lambda_{k_1}^1(t) = \lambda_{k_2}^1(t) = \frac{1}{2}e^{j\omega t}$, then we can get $m_{k_1}^1$ and $m_{k_2}^1$. However, in general $m_k^1 \neq m_{k_1}^1 + m_{k_2}^1$, this suggest for us that the spatial distribution of synapses and inputs is important on neural information processing. In conclusion, the role of synapses can be regarded as a spatio-temporal filter. Figure 2 is the results of simulation of a network of 2000 neurons and the analytic solution for equation (15) and equation (27) in different conditions.

## 4    Summary

In this paper, we deal with the model of the integrate-and-fire neurons with synaptic current dynamics and synaptic depression. In Section 2, first, using the membrane potential equation (1) and combining the synaptic depression equation (2), we derive the evolution equation (4) of the joint distribution density function. Then, we give an approach to cut the evolution equation of the high dimensional function down to one dimension, and get equation (9). Finally, we give the stationary solution and the response of instantaneous fire rate to time-dependence random fluctuation inputs. In Section 3, the analysis and discussion of the model is given and several significant conclusions are presented. This paper can only investigate the IF neuronal model without internal connection. We can also extend to other models, such as the non-linear IF neuronal models of sparsely connected networks of excitatory and inhibitory neurons.

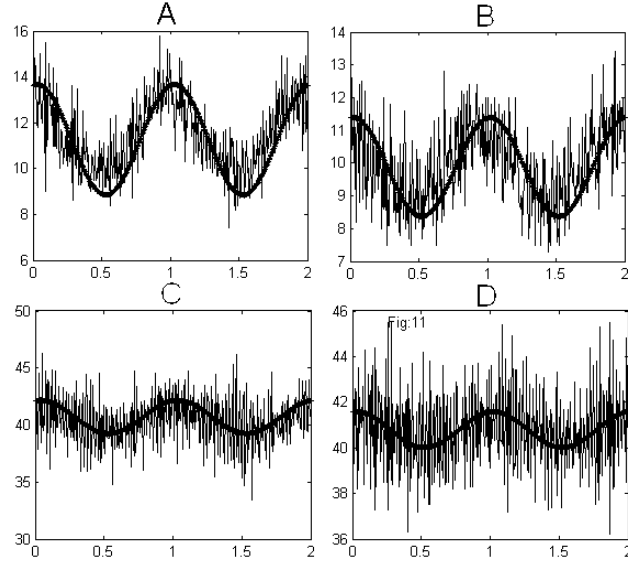

Figure 2: Simulation of a network of 2000 neurons (thin solid line) and the analytic solution (thick solid line) for equation (15) and equation (27), with $\tau_v = 15$(ms), $\tau_d = 1$(s), $A = 0.5$, $U_k = 0.5$, $N = 30$, $\omega = 6.28$(Hz), $\lambda_k^1 = \sin(\omega t)$, $\varepsilon_k \lambda_k^0 = 10$(Hz), $\lambda_k^0 = 70$(Hz) (A and C) and 100(Hz) (B and D), $S_e = 0.5$(A and B) and 0.8(C and D). The horizontal axis is time (0-2s), and the longitudinal axis is the fire rate.

## References

[1] Fourcaud N. & Brunel, N. (2005) Dynamics of the Instantaneous Firing Rate in Response to Changes in Input Statistics. *Journal of Computational Neuroscience* **18**(3):311-321.

[2] Fourcaud, N. & Brunel, N. (2002) Dynamics of the Firing Probability of Noisy Integrate-and-Fire Neurons. *Neural Computation* **14**(9):2057-2110.

[3] Gerstner, W. (2000) Population Dynamics of Spiking Neurons: Fast Transients, Asynchronous States, and Locking. *Neural Computation* **12**(1):43-89.

[4] Silberberg, G., Bethge, M., Markram, H., Pawelzik, K. & Tsodyks, M. (2004) Dynamics of Population Rate Codes in Ensembles of Neocortical Neurons. *J Neurophysiol* **91**(2):704-709.

[5] Abbott, L.F. & Regehr, W.G. (2004) Synaptic Computation. *Nature* **431**(7010):796-803.

[6] Destexhe, A. & Marder, E. (2004) Plasticity in Single Neuron and Circuit Computations. *Nature* **431**(7010):789-795.

[7] Markram, H., Wang, Y. & Tsodyks, M. (1998) Differential Signaling Via the Same Axon of Neocortical Pyramidal Neurons. *Proc Natl Acad Sci USA* **95**(9):5323-5328.
